# SPONTANEOUS AND INFORMATION-TRIGGERED SEGMENTS OF SERIES OF HUMAN BRAIN ELECTRIC FIELD MAPS

D. Lehmann, D. Brandeis*, A. Horst, H. Ozaki* and I. Pal*
Neurology Department, University Hospital, 8091 Zürich, Switzerland

## ABSTRACT

The brain works in a state-dependent manner: processing strategies and access to stored information depends on the momentary functional state which is continuously re-adjusted. The state is manifest as spatial configuration of the brain electric field. Spontaneous and information-triggered brain electric activity is a series of momentary field maps. Adaptive segmentation of spontaneous series into spatially stable epochs (states) exhibited 210 msec mean segments, discontinuous changes. Different maps imply different active neural populations, hence expectedly different effects on information processing: Reaction time differred between map classes at stimulus arrival. Segments might be units of brain information processing (content/mode/step), possibly operationalizing consciousness time. Related units (e.g. triggered by stimuli during figure perception and voluntary attention) might specify brain sub-mechanisms of information treatment.

## BRAIN FUNCTIONAL STATES AND THEIR CHANGES

The momentary functional state of the brain is reflected by the configuration of the brain's electro-magnetic field. The state manifests the strategy, mode, step and content of brain information processing, and the state constrains the choice of strategies and modes and the access to memory material available for processing of incoming information (1). The constraints include the available range of changes of state in PAVLOV's classical "orienting reaction" as response to new or important informations. Different states might be viewed as different functional connectivities between the neural elements.

The orienting reaction (see 1,2) is the result of the first ("pre-attentive") stage of information processing. This stage operates automatically (no involvement of consciousness) and in a parallel mode, and quickly determines whether (a) the information is important or unknown and hence requires increased attention and alertness, i.e. an orienting reaction which means a re-adjustment of functional state in order to deal adequately with the information invoking consciousness for further processing, or whether (b) the information is known or unimportant and hence requires no re-adjustment of state, i.e. that it can be treated further with well-

established ("automatic") strategies. Conscious strategies are slow but flexible (offer wide choice), automatic strategies are fast but rigid.

Examples for functional states on a gross scale are wakefulness, drowsiness and sleep in adults, or developmental stages as infancy, childhood and adolescence, or drug states induced by alcohol or other psychoactive agents. The different states are associated with distinctly different ways of information processing. For example, in normal adults, reality-close, abstracting strategies based on causal relationships predominate during wakefulness, whereas in drowsiness and sleep (dreams), reality-remote, visualizing, associative concatenations of contents are used. Other well-known examples are drug states.

## HUMAN BRAIN ELECTRIC FIELD DATA AND STATES

While alive, the brain produces an ever-changing electromagnetic field, which very sensitively reflects global and local states as effected by spontaneous activity, incoming information, metabolism, drugs, and diseases. The electric component of the brain's electromagnetic field as non-invasively measured from the intact human scalp shows voltages between 0.1 and 250 microVolts, temporal frequencies between 0.1 and 30, 100 or 3000 Hz depending on the examined function, and spatial frequencies up to 0.2 cycles/cm.

Brain electric field data are traditionally viewed as time series of potential differences between two scalp locations (the electroencephalogram or EEG). Time series analysis has offered an effective way to class different gross brain functional states, typically using EEG power spectral values. Differences between power spectra during different gross states typically are greater than between different locations. States of lesser functional complexity such as childhood vs adult states, sleep vs wakefulness, and many drug-states vs non-drug states tend to increased power in slower frequencies (e.g. 1,4).

Time series analyses of epochs of intermediate durations between 30 and 10 seconds have demonstrated (e.g. 1,5,6) that there are significant and reliable relations between spectral power or coherency values of EEG and characteristics of human mentation (reality-close thoughts vs free associations, visual vs non-visual thoughts, positive vs negative emotions).

Viewing brain electric field data as series of momentary field maps (7,8) opens the possibility to investigate the temporal microstructure of brain functional states in the sub-second range. The rationale is that the momentary configuration of activated neural elements represents a given brain functional state, and that the spatial pattern of activation is reflected by the momentary brain electric field which is recordable on the scalp as a momentary field map. Different configurations of activation (different field maps) are expected to be associated with different modes, strategies, steps and contents of information processing.

## SEGMENTATION OF BRAIN ELECTRIC MAP SERIES INTO STABLE SEGMENTS

When viewing brain electric activity as series of maps of
momentary potential distributions, changes of functional state are
recognizable as changes of the "electric landscapes" of these maps.
Typically, several successive maps show similar landscapes, then
quickly change to a new configuration which again tends to persist
for a number of successive maps, suggestive of stable states
concatenated by non-linear transitions (9,10). Stable map landscapes
might be hypothesized to indicate the basic building blocks of
information processing in the brain, the "atoms of thoughts". Thus,
the task at hand is the recognition of the landscape configurations;
this leads to the adaptive segmentation of time series of momentary
maps into segments of stable landscapes during varying durations.

We have proposed and used a method which describes the
configuration of a momentary map by the locations of its maximal and
minimal potential values, thus invoking a dipole model. The goal
here is the phenomenological recognition of different momentary
functional states using a very limited number of major map features
as classifiers, and we suggest conservative interpretion of the data
as to real brain locations of the generating processes which always
involve millions of neural elements.

We have studied (11) map series recorded from 16 scalp locations
over posterior skull areas from normal subjects during relaxation
with closed eyes. For adaptive segmentation, the maps at the times
of maximal map relief were selected for optimal signal/noise
conditions. The locations of the maximal and minimal (extrema)
potentials were extracted in each map as descriptors of the
landscape; taking into account the basically periodic nature of
spontaneous brain electric activity (Fig. 1), extrema locations were
treated disregarding polarity information. If over time an extreme
left its pre-set spatial window (say, one electrode distance), the
segment was terminated. The map series showed stable map
configurations for varying durations (Fig. 2), and discontinuous,
step-wise changes. Over 6 subjects, resting alpha-type EEG showed
210 msec mean segment duration; segments longer than 323 msec
covered 50% of total time; the most prominent segment class (1.5% of
all classes) covered 20% of total time (prominence varied strongly
over classes; not all possible classes occurred). Spectral power and
phase of averages of adaptive and pre-determined segments
demonstrated the adequacy of the strategy and the homogeneity of
adaptive segment classes by their reduced within-class variance.
Segmentation using global map dissimilarity (sum of Euklidian
difference vs average reference at all measured points) emulates the
results of the extracted-characteristics-strategy.

## FUNCTIONAL SIGNIFICANCE OF MOMENTARY MICRO STATES

Since different maps of momentary EEG fields imply activity of
different neural populations, different segment classes must
manifest different brain functional states with expectedly different

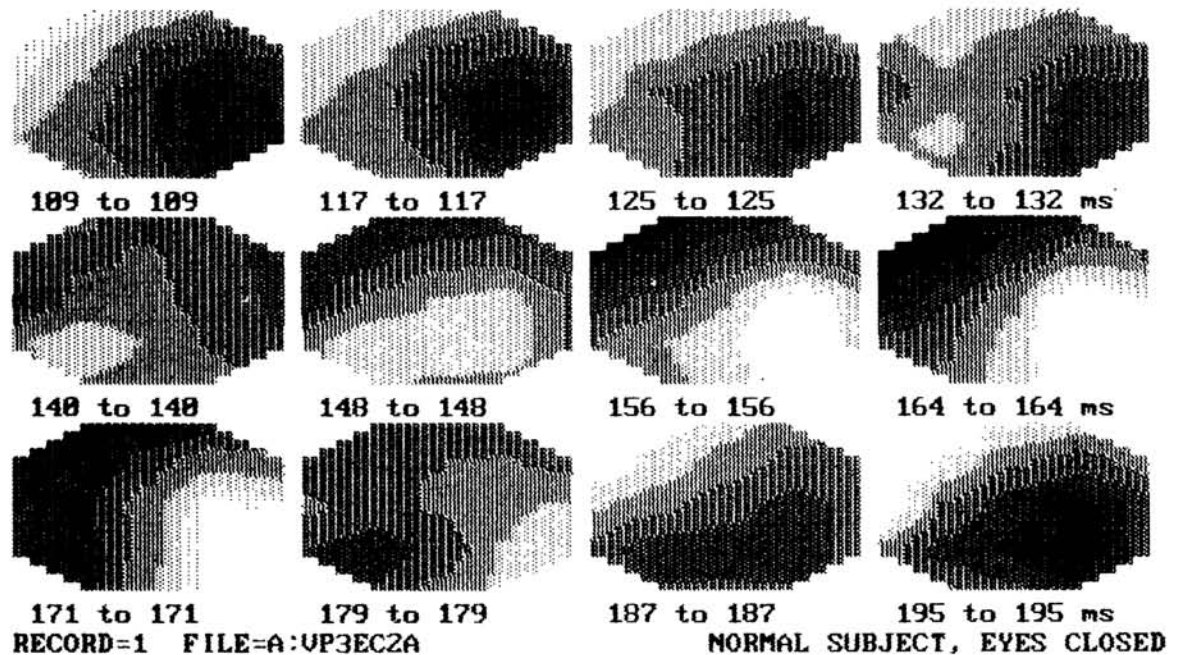

109 to 109    117 to 117    125 to 125    132 to 132 ms

140 to 140    148 to 148    156 to 156    164 to 164 ms

171 to 171    179 to 179    187 to 187    195 to 195 ms

RECORD=1   FILE=A:VP3EC2A           NORMAL SUBJECT, EYES CLOSED

Fig. 1. Series of momentary potential distribution maps of the brain field recorded from the scalp of a normal human during relaxation with closed eyes. Recording with 21 electrodes (one 5-electrode row added to the 16-electrode array in Fig. 2) using 128 samples/sec/ channel. Head seen from above, left ear left; white positive, dark negative, 8 levels from +32 to -32 microVolts. Note the periodic reversal of field polarity within the about 100 msec (one cycle of the 8-12Hz so-called "EEG alpha" activity) while the field configuration remains largely constant. – This recording and display was done with a BRAIN ATLAS system (BioLogic Systems, Mundelein, IL).

effects on ongoing information processing. This was supported by measurements of selective reaction time to acoustic stimuli which were randomly presented to eight subjects during different classes of EEG segments (323 responses for each subject). We found significant reaction time differences over segment classes (ANOVA p smaller than .02), but similar characteristics over subjects. This indicates that the momentary sub-second state as manifest in the potential distribution map significantly influences the behavioral consequence of information reaching the brain.

Presentation of information is followed by a sequence of potential distribution maps ("event-related potentials" or ERP's, averaged over say, 100 presentations of the same stimulus, see 12). The different spatial configurations of these maps (12) are thought to reflect the sequential stages of information processing associated with "components" of event-related brain activity (see e.g. 13) which are traditionally defined as times of maximal voltages after information input (maximal response strength).

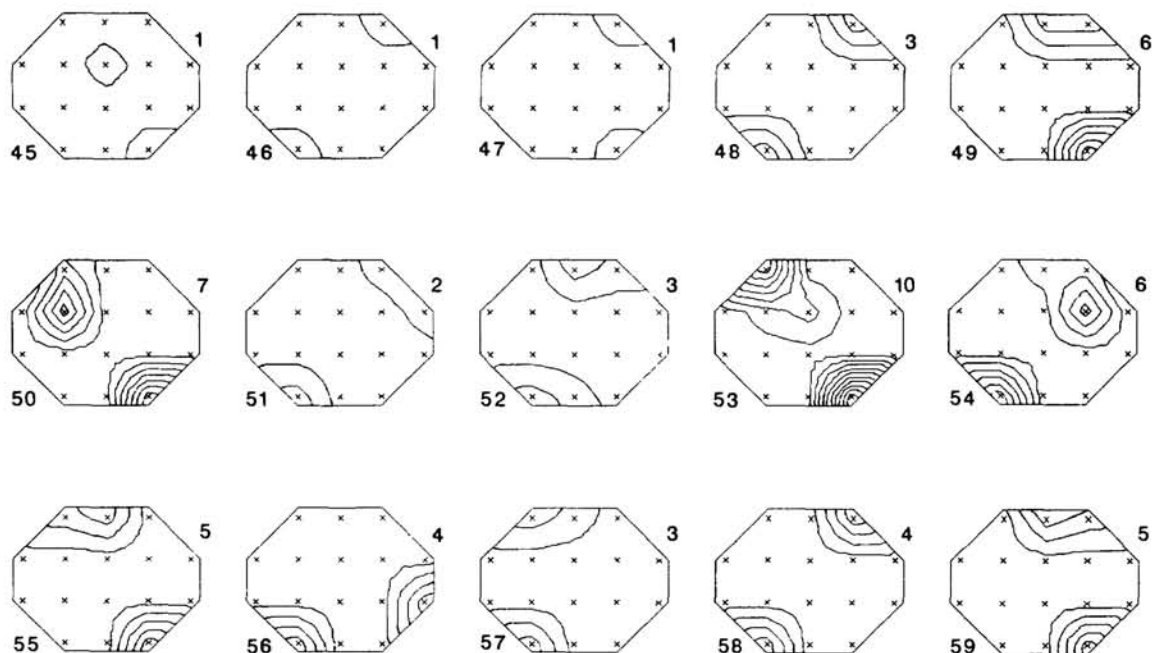

Fig. 2. Sequence of spatially stable segments during a spontaneous series of momentary EEG maps of 3.1 sec duration in a normal volunteer. Each map shows the occurrence of the extreme potential values during one adaptively determined segment: the momentary maps were searched for the locations of the two extreme potentials; these locations were accumulated, and linearly interpolated between electrodes to construct the present maps. (The number of iso-frequency-of-occurrence lines therefore is related to the number of searched maps). - Head seen from above, left ear left, electrode locations indicated by crosses, most forward electrode at vertex. Data FIR filtered to 8-12Hz (alpha EEG). The figure to the left below each map is a running segment number. The figure to the right above each map multiplied by 50 indicates the segment duration in msec.

Application of the adaptive segmentation procedure described above for identification of functional components of event-related brain electric map sequences requires the inclusion of polarity information (14); such adaptive segmentation permits to separate different brain functional states without resorting to the strength concept of processing stages.

An example (12) might illustrate the type of results obtained with this analysis: Given segments of brain activity which were triggered by visual information showed different map configurations when subjects paid attention vs when they paid no attention to the stimulus, and when they viewed figures vs meaningless shapes as

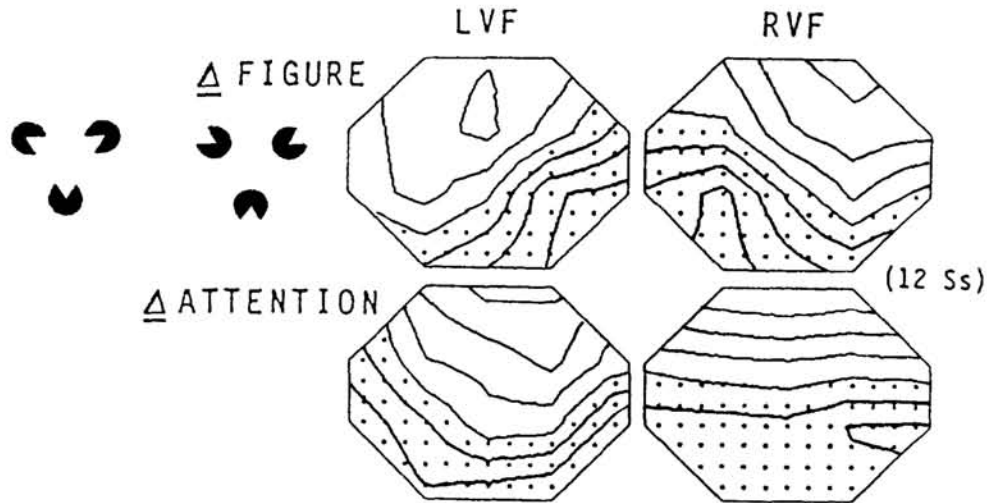

Fig. 3. Four difference maps, computed as differences between maps
obtained during (upper row) perception of a visual "illusionary"
triangle figure (left picture) minus a visual non-figure (right)
shown to the left and right visual hemi-fields (LVF, RVF), and
obtained during (lower row) attending minus during ignoring the
presented display. The analysed segment covered the time from 168 to
200 msec after stimulus presentations. - Mean of 12 subjects. Head
seen from above, left ear left, 16 electrodes as in Fig. 2,
isopotential contour lines at 0.1 microVolt steps, dotted negative
referred to mean of all values. The "illusionary" figure stimulus
was studied by Kanisza (16); see also (12). - Note that the mirror
symmetric configuration of the difference maps for LVF and RVF is
found for the "figure" effect only, not for the "attention" effect,
but that the anterior-posterior difference is similar for both cases.

stimuli. Fig. 3 illustrates such differences in map configuration.
The "attention"-induced and "figure"-induced changes in map
configuration showed certain similarities e.g. in the illustrated
segment 168-200 msec after information arrival, supporting the
hypothesis that brain mechanisms for figure perception draw on brain
resources which in other circumstances are utilized in voluntary
attention.
   The spatially homogeneous temporal segments might be basic
building blocks of brain information processing, possibly
operationalizing consciousness time (15), and offering a common
concept for analysis of brain spontaneous activity and event related
brain potentials. The functional significance of the segments might
be types/ modes/ steps of brain information processing or
performance. Identification of related building blocks during
different brain functions accordingly could specify brain sub-
mechanisms of information treatment.

Acknowledgement: Financial support by the Swiss National Science Foundation (including Fellowships to H.O. and I.P.) and by the EMDO, the Hartmann Muller and the SANDOZ Foundation is gratefully acknowledged.

## Footnotes

* Present addresses: D.B. at Psychiat. Dept., V.A. Med. Center, San Francisco CA 94121; H.O. at Lab. Physiol. for the Developmentally Handicapped, Ibaraki Univ., Mito, Japan 310; I.P. at BioLogic Systems Corp., Mundelein IL 60060.

## REFERENCES

1. M. Koukkou and D. Lehmann, Brit. J. Psychiat. 142, 221-231 (1983).
2. A. Ohman, In: H.D. Kimmel, E.H. von Olst and J.F. Orlebeke (Eds.), Drug-Discrimination and State Dependent Learning (Academic Press, New York, 1979), pp. 283-318.
3. A. Katada, H. Ozaki, H. Suzuki and K. Suhara, Electroenceph. Clin. Neurophysiol. 52, 192-201 (1981).
4. M. Koukkou and D. Lehmann, Biol. Psychiat. 11, 663-677 (1976).
5. J. Berkhout, D.O. Walter and W.R. Adey, Electroenceph. clin. Neurophysiol. 27, 457-469 (1969).
6. P. Grass, D. Lehmann, B. Meier, C.A. Meier and I. Pal, Sleep Res. 16, 231 (1987).
7. D. Lehmann, Electroenceph. Clin. Neurophysiol. 31, 439-449 (1971).
8. D. Lehmann, In: H.H. Petsche and M.A.B. Brazier (eds.), Synchronization of EEG Activity in Epilepsies (Springer, Wien, 1972), pp. 307-326.
9. H. Haken, Advanced Synergetics (Springer, Heidelberg, 1983).
10. J.J. Wright, R.R. Kydd and G.L. Lees, Biol. Cybern., 1985, 53, 11-17.
11. D. Lehmann, H. Ozaki and I. Pal, Electroenceph. Clin. Neurophysiol. 67, 271-288 (1987).
12. D. Brandeis and D. Lehmann, Neuropsychologia 24, 151-168 (1986).
13. A.S. Gevins, N.H. Morgan, S.L. Bressler, B.A. Cutillo, R.M. White, J. Illes, D.S. Greer, J.C.Doyle and M. Zeitlin, Science 235, 580-585 (1987).
14. D. Lehmann and W. Skrandies, Progr. Neurobiol. 23, 227-250 (1984).
15. B. Libet, Human Neurobiol. 1, 235-242 (1982).
16. G. Kanisza, Organization of Vision (Praeger, New York, 1979).
